# A BACK–PROPAGATION ALGORITHM WITH OPTIMAL USE OF HIDDEN UNITS

Yves Chauvin
Thomson–CSF, Inc
(and Psychology Department, Stanford University)
630, Hansen Way (Suite 250)
Palo Alto, CA 94306

## ABSTRACT

This paper presents a variation of the back–propagation algorithm that makes optimal use of a network hidden units by decreasing an "energy" term written as a function of the squared activations of these hidden units. The algorithm can automatically find optimal or nearly optimal architectures necessary to solve known Boolean functions, facilitate the interpretation of the activation of the remaining hidden units and automatically estimate the complexity of architectures appropriate for phonetic labeling problems. The general principle of the algorithm can also be adapted to different tasks: for example, it can be used to eliminate the [0, 0] local minimum of the [−1, +1] logistic activation function while preserving a much faster convergence and forcing binary activations over the set of hidden units.

## PRINCIPLE

This paper describes an algorithm which makes optimal use of the hidden units in a network using the standard back–propagation algorithm (Rumelhart, Hinton & Williams, 1986). Optimality is defined as the minimization of a function of the "energy" spent by the hidden units throughtout the network, independently of the chosen architecture, and where the energy is written as a function of the squared activations of the hidden units.

The standard back–propagation algorithm is a gradient descent algorithm on the following cost function:

$$C = \sum_{j}^{P} \sum_{i}^{O} (d_{ij} - o_{ij})^2 \qquad [1]$$

where $d$ is the desired output of an output unit, $o$ the actual output, and where the sum is taken over the set of output units $O$ for the set of training patterns $P$.

The following algorithm implements a gradient descent on the following cost function:

$$C = \mu_{er} \sum_{j}^{P} \sum_{i}^{O} (d_{ij} - o_{ij})^2 + \mu_{en} \sum_{j}^{P} \sum_{i}^{H} e(o_{ij}^2) \qquad [2]$$

where $e$ is a positive monotonic function and where the sum of the second term is now taken over a set or subset of the hidden units $H$. The first term of this cost function will be called the error term, the second, the energy term.

In principle, the theoretical minimum of this function is found when the desired activations are equal to the actual activations for all output units and all presented patterns and when the hidden units do not "spend any energy". In practical cases, such a minimum cannot be reached and the hidden units have to "spend some energy" to solve a given problem. The quantity of energy will be in part determined by the relative importance given to the error and energy terms during gradient descent. In principle, if a hidden unit has a constant activation whatever the pattern presented to the network, it contributes to the energy term only and will be "suppressed" by the algorithm. The precise energy distribution among the hidden units will depend on the actual energy function $e$.

## ANALYSIS

## ALGORITHM IMPLEMENTATION

We can write the total cost function that the algorithm tries to minimize as a weighted sum of an error and energy term:

$$C = \mu_{er}E_{er} + \mu_{en}E_{en} \qquad [3]$$

The first term is the error term used with the standard back–propagation algorithm in Rumelhart et al. If we have $h$ hidden layers, we can write the total energy term as a sum of all the energy terms corresponding to each hidden layer:

$$E_{en} = \sum_{i}^{h} \sum_{j}^{H_i} e(o_j^2) \qquad [4]$$

To decrease the energy of the uppermost hidden layer $H_h$, we can compute the derivative of the energy function with respect to the weights. This derivative will be null for any weight "above" the considered hidden layer. For any weight just below the considered hidden layer, we have (using Rumelhart et al. notation):

$$\frac{\partial E_{en}}{\partial w_{ij}} = \frac{\partial E_{en}}{\partial net_i} \frac{\partial net_i}{\partial w_{ij}} = \delta_i^{en} \frac{\partial net_i}{\partial w_{ij}} = \delta_i^{en} o_j \qquad [5]$$

$$\delta_i^{en} = \frac{\partial e(o_i^2)}{\partial net_i} = \frac{\partial e(o_i^2)}{\partial o_i^2} \frac{\partial o_i^2}{\partial o_i} \frac{\partial o_i}{\partial net_i} = 2e' o_i f'_i(net_i) \qquad [6]$$

where the derivative of $e$ is taken with respect to the "energy" of the unit $i$ and where $f$ corresponds to the logistic function. For any hidden layer below the considered layer $h$, the chain rule yields:

$$\delta_k^{en} = f'_k(net_k) \sum_j \delta_j^{en} w_{jk} \qquad [7]$$

This is just standard back–propagation with a different back–propagated term. If we minimize both the error at the output layer and the energy of the hidden layer $h$, we can compute the complete weight change for any connection below layer $h$:

$$\Delta w_{kl} = -a\mu_{er}\delta_k^{er} o_l - a\mu_{en}\delta_k^{en} o_l = -ao_l(\mu_{er}\delta_k^{er} + \mu_{en}\delta_k^{en}) = -ao_l\delta_k^{ac} \qquad [8]$$

where $\delta_k^{ac}$ is now the delta accumulated for error and energy that we can write as a function of the deltas of the upper layer:

$$\delta_k^{ac} = f'_k(net_k) \sum_i (\mu_{er}\delta_i^{er} + \mu_{en}\delta_i^{en}) w_{ik} = f'_k(net_k) \sum_i \delta_i^{ac} w_{ik} \qquad [9]$$

This means that instead of propagating the delta for both energy and error, we can compute an accumulated delta for hidden layer $h$ and propagate it back throughout the network. If we minimize the energy of the layers $h$ and $h-1$, the new accumulated delta will equal the previously accumulated delta added to a new delta energy on layer $h-1$. The procedure can be repeated throughout the complete network. In short, the back–propagated error signal used to change the weights of each layer is simply equal to the back–propagated signal used in the previous layer augmented with the delta energy of the current hidden layer. (The algorithm is local and easy to implement).

## ENERGY FUNCTION

The algorithm is sensitive to the energy function $e$ being minimized. The functions used in the simulations described below have the following derivative with

respect to the squared activations/energy (only this derivative is necessary to implement the algorithm, see Equation [6]):

$$e' = \frac{\partial e(o^2)}{\partial o_i^2} = \frac{1}{(1+o^2)^n} \qquad [10]$$

where $n$ is an integer that determines the precise shape of the energy function (see Table 1) and modulates the behavior of the algorithm in the following way. For $n = 0$, $e$ is a linear function of the energy: "high and low energy" units are equally penalized. For $n = 1$, $e$ is a logarithmic function and "low energy" units become more penalized than "high energy" units, in proportion to the linear case. For $n = 2$, the energy penalty may reach an asymptote as the energy increases: "high energy" units are not penalized more than "middle energy" units. In the simulations, as expected, it appears that higher values of $n$ tend to suppress "low energy" units. (For $n > 2$, the behavior of the algorithm was not significantly different from $n = 2$, for the tests described below).

TABLE 1: Energy Functions.

| $n$ | $0$ | $1$ | $2$ | $n>2$ |
|---|---|---|---|---|
| $e$ | $o^2$ | $Log(1+o^2)$ | $\dfrac{o^2}{1+o^2}$ | ? |

## BOOLEAN EXAMPLES

The algorithm was tested with a set of Boolean problems. In typical tasks, the energy of the network significantly decreases during early learning. Later on, the network finds a better minimum of the total cost function by decreasing the error and by "spending" energy to solve the problem. Figure 1 shows energy and error in function of the number of learning cycles during a typical task (XOR) for 4 different runs. For a broad range of the energy learning rate, the algorithm is quite stable and finds the solution to the given problem. This nice behavior is also quite independent of the variations of the onset of energy minimization.

## EXCLUSIVE OR AND PARITY

The algorithm was tested with EXOR for various network architectures. Figure 2 shows an example of the activation of the hidden units after learning. The algorithm finds a minimal solution (2 hidden units, "minimum logic") to solve the XOR problem when the energy is being minimized. This minimal solution is actually found whatever the starting number of hidden units. If several layers are used, the algorithm finds an optimal or nearly-optimal size for each layer.

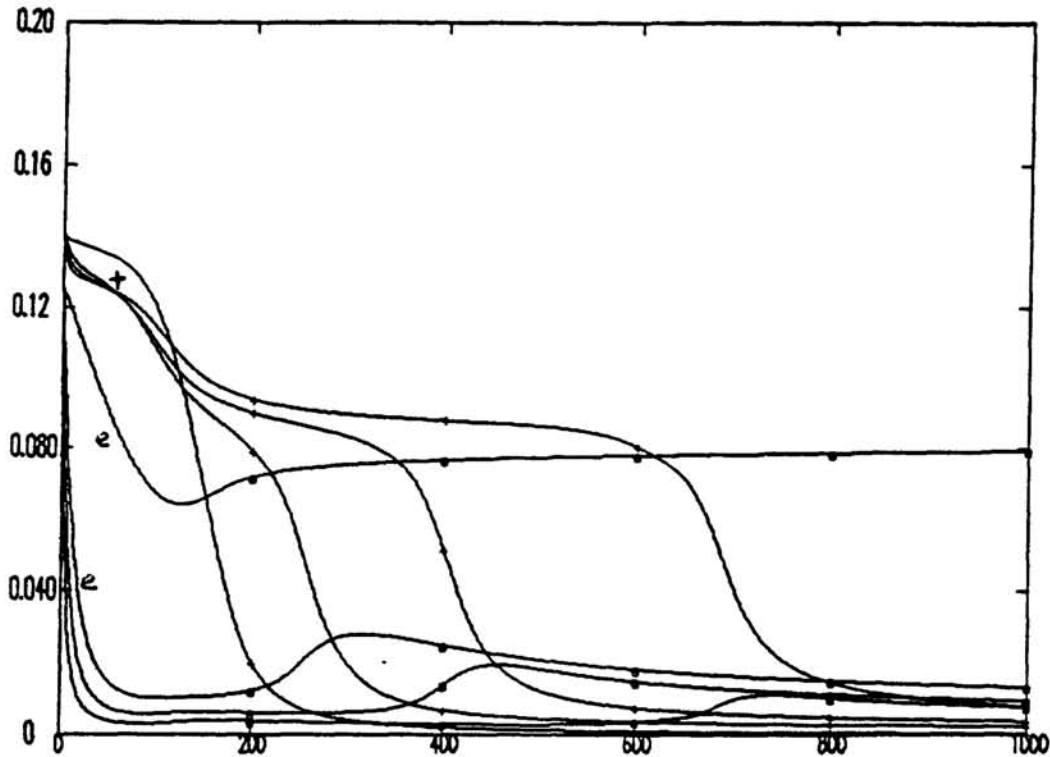

**Figure 1.** Energy and error curves as a function of the number of pattern presentations for different values of the "energy" rate (0, .1, .2, .4). Each energy curve ("e" label) is associated with an error curve ("+" label). During learning, the units "spend" some energy to solve the given task.

With parity 3, for a [−1, +1] activation range of the sigmoid function, the algorithm does not find the 2 hidden units optimal solution but has no problem finding a 3 hidden units solution, independently of the starting architecture.

## SYMMETRY

The algorithm was tested with the symmetry problem, described in Rumelhart et al. The minimal solution for this task uses 2 hidden units. The simplest form of the algorithm does not actually find this minimal solution because some weights from the hidden units to the output unit can actually grow enough to compensate the low activations of the hidden units. However, a simple weight decay can prevent these weights from growing too much and allows the network to find the minimal solution. In this case, the total cost function being minimized simply becomes:

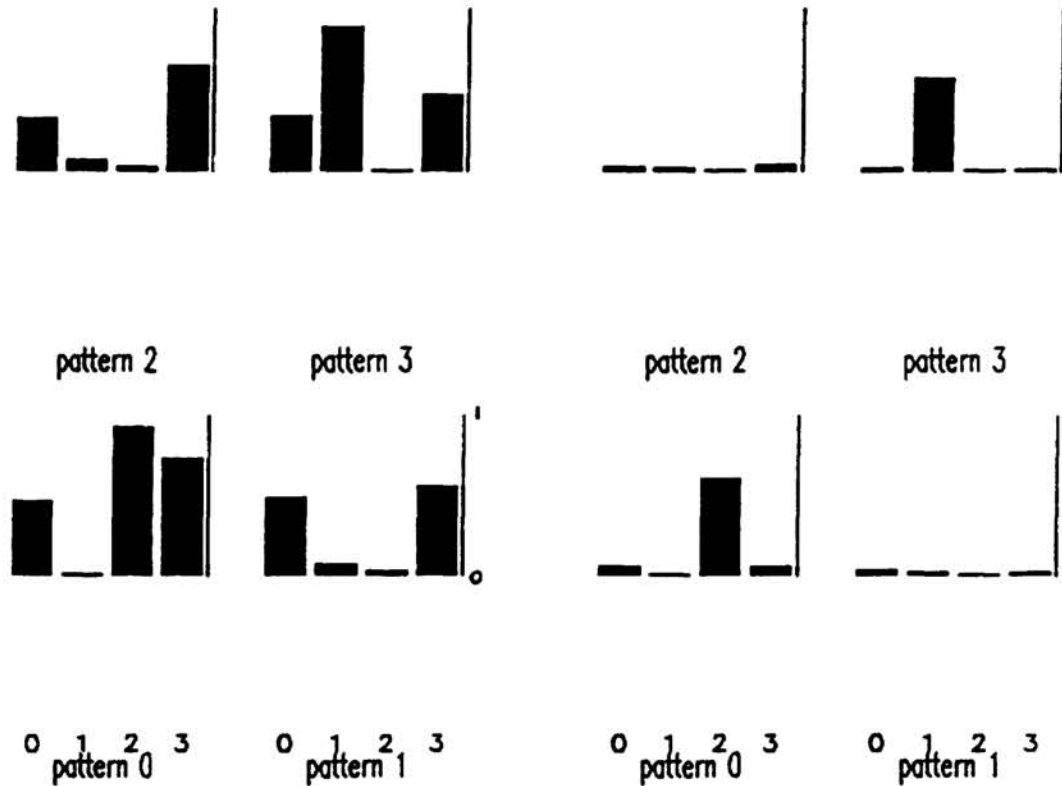

**Figure 2.** Hidden unit activations of a 4 hidden unit network over the 4 EXOR patterns when (left) standard back–propagation and (right) energy minimization are being used during learning. The network is reduced to minimal size (2 hidden units) when the energy is being minimized.

$$C = \mu_{er} \sum_{j}^{P} \sum_{i}^{O} (d_{ij} - o_{ij})^2 + \mu_{en} \sum_{j}^{P} \sum_{i}^{H} e(o_{ij}^2) + \mu_{w} \sum_{ij}^{W} w_{ij}^2 \qquad [11]$$

## PHONETIC LABELING

The algorithm was tested with a phonetic labeling task. The input patterns consisted of spectrograms (single speaker, 10x3.2ms spaced time frames, centered, 16 frequencies) corresponding to 9 syllables [ba], [da], [ga], [bi], [di], [gi], and [bu], [du], [gu]. The task of the network was to classify these spectrograms (7 tokens per syllable) into three categories corresponding to the three consonants [b], [g], and [g]. Starting with 12 hidden units, the algorithm reduced the network to 3 hidden units. (A hidden unit is considered unused when its activation over the entire range of patterns contributes very little to the activations of the output units). With standard back–propagation, all of the 12 hidden units are usually being used. The resulting network is consistent with the sizes of the hidden layers used by Elman and Zipser (1986) for similar tasks.

## EXTENSION OF THE ALGORITHM

Equation [2] represents a constraint over the set of possible LMS solutions found by the back–propagation algorithm. With such a constraint, the "zero–energy" level of the hidden units can be (informally) considered as an attractor in the solution space. However, by changing the sign of the energy gradient, such a point now constitutes a repellor in this space. Having such repellors might be useful when a set of activation values are to be avoided during learning. For example, if the activation range of the sigmoid transfer function is [−1, +1], the learning speed of the back–propagation algorithm can be greatly improved but the [0, 0] unit activation point (zero–input, zero–output) often behaves as a local minimum. By inversing the sign of the energy gradient during early learning, it is possible to have the point [0, 0] act as a repellor, forcing the network to make "maximal use" of its resources (hidden units). This principle was tested on the parity–3 problem with a network of 7 hidden units. For a given set of coefficients, standard back–propagation can solve parity–3 in about 15 cycles but yields about 65% of local minima in [0, 0]. By using the "repulsion" constraint, parity–3 can be solved in about 20 cycles with 0% of local minima.

Interestingly, it is also possible to design a "trajectory" of such constraints during learning. For example, the [0, 0] activation point can be built as a repellor during early learning in order to avoid the corresponding local minimum, then as an attractor during middle learning to reduce the size of the hidden layer, and as a repulsor during late learning, to force the hidden units to have binary activations. This type of trajectory was tested on the parity–3 problem with 7 hidden units. In this case, the algorithm always avoids the [0, 0] local minimum. Moreover, the network can be reduced to 3 or 4 hidden units taking binary values over the set of input patterns. In contrast, standard back–propagation often gets stuck in local minima and uses the initial 7 hidden units with analog activation values.

## CONCLUSION

The present algorithm simply imposes a constraint over the LMS solution space. It can be argued that limiting such a solution space can in some cases increase the generalizing properties of the network (curve–fitting analogy). Although a complete theory of generalization has yet to be formalized, the present algorithm presents a step toward the automatic design of "minimal" networks by imposing constraints on the activations of the hidden units. (Similar constraints on weights can be imposed and have been tested with success by D. E. Rumelhart, Personal Communication. Combinations of constraints on weights and activations are being tested). What is simply shown here is that this energy minimization principle is easy to implement, is robust to a broad range of parameter values, can find minimal or nearly optimal network sizes when tested with a variety of tasks and can be used to "bend" trajectories of activations during learning.

**Ackowledgments**

This research was conducted at Thomson–CSF, Inc. in Palo Alto. I would like to thank the Thomson neural net team for useful discussions. Dave Rumelhart and the PDP team at Stanford University were also very helpful. I am especially greateful to Yoshiro Miyata, from Bellcore, for having letting me use his neural net simulator (SunNet) and to Jeff Elman, from UCSD, for having letting me use the speech data that he collected.

**References.**

J. L. Elman & D. Zipser. Learning the hidden structure of speech. ICS Technical Report 8701. Institute for Cognitive Science. University of California, San Diego (1987).

D. E. Rumelhart, G. E. Hinton & R. J. Williams. Learning internal representaions by error propagation. In D. E. Rumelhart & J. L. McClelland (Eds.), *Parallel Distributed Processing: Exploration in the Microstructure of Cognition. Vol. 1.* Cambridge, MA: MIT Press/ Bradford Books (1986).